# Semi-supervised Regression using Hessian Energy with an Application to Semi-supervised Dimensionality Reduction

**Kwang In Kim**[1]**, Florian Steinke**[2,3]**, and Matthias Hein**[1]
[1]Department of Computer Science, Saarland University Saarbrücken, Germany
[2]Siemens AG Corporate Technology Munich, Germany
[3]MPI for Biological Cybernetics, Germany
`{kimki,hein}@cs.uni-sb.de,Florian.Steinke@siemens.com`

## Abstract

Semi-supervised regression based on the graph Laplacian suffers from the fact that the solution is biased towards a constant and the lack of extrapolating power. Based on these observations, we propose to use the second-order Hessian energy for semi-supervised regression which overcomes both these problems. If the data lies on or close to a low-dimensional submanifold in feature space, the Hessian energy prefers functions whose values vary linearly with respect to geodesic distance. We first derive the Hessian energy for smooth manifolds and continue to give a stable estimation procedure for the common case where only samples of the underlying manifold are given. The preference of ''linear'' functions on manifolds renders the Hessian energy particularly suited for the task of semi-supervised dimensionality reduction, where the goal is to find a user-defined embedding function given some labeled points which varies smoothly (and ideally linearly) along the manifold. The experimental results suggest superior performance of our method compared with semi-supervised regression using Laplacian regularization or standard supervised regression techniques applied to this task.

## 1 Introduction

Central to semi-supervised learning is the question how unlabeled data can help in either classification or regression. A large class of methods for semi-supervised learning is based on the *manifold assumption*, that is, the data points do not fill the whole feature space but they are concentrated around a low-dimensional submanifold. Under this assumption unlabeled data points can be used to build adaptive regularization functionals which penalize variation of the regression function only *along* the underlying manifold.

One of the main goals of this paper is to propose an appropriate regularization functional on a manifold, the Hessian energy, and show that it has favourable properties for semi-supervised regression compared to the well known Laplacian regularization [2, 12]. Opposite to the Laplacian regularizer, the Hessian energy allows functions that extrapolate, i.e. functions whose values are not limited to the range of the training outputs. Particularly if only few labeled points are available, we show that this extrapolation capability leads to significant improvements. The second property of the proposed Hessian energy is that it favors functions which vary linearly along the manifold, so-called *geodesic functions* defined later. By linearity we mean that the output values of the functions change linearly along geodesics in the input manifold. This property makes it particularly useful as a tool for semi-supervised dimensionality reduction [13], where the task is to construct user-defined embeddings based on a given subset of labels. These user-guided embeddings are supposed to vary smoothly or even linearly along the manifold, where the later case corresponds to a setting where the user tries to

recover a low-distortion parameterization of the manifold. Moreover, due to user defined labels the interpretability of the resulting parameterization is significantly improved over unsupervised methods like Laplacian [1] or Hessian [3] eigenmaps. The proposed Hessian energy is motivated by the recently proposed Eells energy for mappings between manifolds [11], which contains as a special case the regularization of real-valued functions on a manifold. In flavour, it is also quite similar to the operator constructed in Hessian eigenmaps [3]. However, we will show that their operator due to problems in the estimation of the Hessian, leads to useless results when used as regularizer for regression. On the contrary, our novel estimation procedure turns out to be more stable for regression and as a side effects leads also to a better estimation of the eigenvectors used in Hessian eigenmaps.

We present experimental results on several datasets, which show that our method for semi-supervised regression is often superior to other semi-supervised and supervised regression techniques.

## 2   Regression on manifolds

Our approach for regression is based on regularized empirical risk minimization. First, we will discuss the problem and the regularizer in the ideal case where we know the manifold exactly, corresponding to the case where we have access to an unlimited number of unlabeled data. In the following we denote by $M$ the $m$-dimensional data-submanifold in $\mathbb{R}^d$. The supervised regression problems for a training set of $l$ points $(X_i, Y_i)_{i=1}^l$ can then be formulated as,

$$\operatorname*{arg\,min}_{f \in C^\infty(M)} \frac{1}{l} \sum_{i=1}^l L(Y_i, f(X_i)) + \lambda\, S(f),$$

where $C^\infty(M)$ is the set of smooth functions on $M$, $L : \mathbb{R} \times \mathbb{R} \to \mathbb{R}$ is the loss function and $S : C^\infty(M) \to \mathbb{R}$ is the regularization functional. For simplicity we use the squared loss $L(y, f(x)) = (y - f(x))^2$, but the framework can be easily extended to other convex loss functions.

Naturally, we do not know the manifold $M$ the data is lying on. However, we have unlabeled data which can be used to estimate it, or more precisely we can use the unlabeled data to build an estimate $\hat{S}(f)$ of the true regularizer $S(f)$. The proper estimation of $S(f)$ will be the topic of the next section. For the moment we just want to discuss regularization functionals in the ideal case, where we know the manifold. However, we would like to stress already here that for our framework to work it does not matter if the data lies on or close to a low-dimensional manifold. Even the dimension can change from point to point. The only assumption we make is that the data generating process does not fill the whole space but is concentrated on a low-dimensional structure.

**Regularization on manifolds.**   Our main goal is to construct a regularization functional on manifolds, which is particularly suited for semi-supervised regression and semi-supervised dimensionality reduction. We follow here the framework of [11] who discuss regularization of mappings between manifolds, where we are interested in the special case of real-valued output. They propose to use the so called *Eells*-energy $S_{\text{Eells}}(f)$, which can be written for real-valued functions, $f : M \to \mathbb{R}$, as,

$$S_{\text{Eells}}(f) = \int_M \|\nabla_a \nabla_b f\|^2_{T_x^* M \otimes T_x^* M}\, dV(x),$$

where $\nabla_a \nabla_b f$ is the second covariant derivative of $f$ and $dV(x)$ is the natural volume element, see [7]. Note, that the energy is by definition independent of the coordinate representation and depends only on the properties of $M$. For details we refer to [11]. This energy functional looks quite abstract. However, in a special coordinate system on $M$, the so called normal coordinates, one can evaluate it quite easily. In sloppy terms, normal coordinates at a given point $p$ are coordinates on $M$ such that the manifold looks as Euclidean as possible (up to second order) around $p$. Thus in normal coordinates $x_r$ centered at $p$,

$$\nabla_a \nabla_b f\Big|_p = \sum_{r,s=1}^m \frac{\partial^2 f}{\partial x_r \partial x_s}\Big|_p dx_a^r \otimes dx_b^s \quad \Longrightarrow \quad \|\nabla_a \nabla_b f\|^2_{T_p^* M \otimes T_p^* M} = \sum_{r,s=1}^m \left(\frac{\partial^2 f}{\partial x_r \partial x_s}\right)^2, \quad (1)$$

so that at $p$ the norm of the second covariant derivative is just the Frobenius norm of the Hessian of $f$ in normal coordinates. Therefore we call the resulting functional the *Hessian* regularizer $S_{\text{Hess}}(f)$.

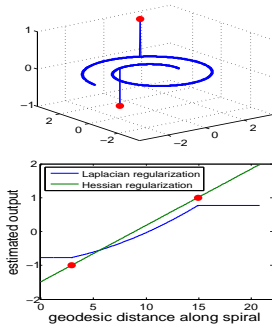

Figure 1: Difference between semi-supervised regression using Laplacian and Hessian regularization for fitting two points on the one-dimensional spiral. The Laplacian regularization has always a bias towards the constant function (for a non-zero regularization parameter it will not fit the data exactly) and the extrapolation beyond data points to the boundary of the domain is always constant. The non-linearity of the fitted function between the data point arises due to the non-uniform sampling of the spiral. On the contrary the Hessian regularization fits the data perfectly and extrapolates nicely to unseen data, since it's null space contains functions which vary linearly with the geodesic distance.

Before we discuss the discretization, we would like to discuss some properties of this regularizer. In particular, its difference to the regularizer $S_\Delta(f)$ using the Laplacian,

$$S_\Delta(f) = \int_M \|\nabla f\|^2 \, dV(x)$$

proposed by Belkin and Niyogi [2] for semi-supervised classification and in the meantime also adopted for semi-supervised regression [12]. While this regularizer makes sense for classification, it is of limited use for regression. The problem is that the null space $N_S = \{f \in C^\infty(M) \,|\, S(f) = 0\}$ of $S_\Delta$, that is the functions which are not penalized, are only the constant functions on $M$. The following adaptation of a result in [4] shows that the Hessian regularizer has a richer null-space.

**Proposition 1 (Eells, Lemaire [4])** *A function $f : M \to \mathbb{R}$ with $f \in C^\infty(M)$ has zero second derivative, $\nabla_a \nabla_b f\big|_x = 0, \quad \forall x \in M$, if and only if for **any** geodesic $\gamma : (-\varepsilon, \varepsilon) \to M$ parameterized by arc length $s$, there exists a constant $c_\gamma$ depending only on $\gamma$ such that*

$$\frac{\partial}{\partial s} f\big(\gamma(s)\big) = c_\gamma, \quad \forall \, -\varepsilon < s < \varepsilon.$$

We call functions $f$ which fulfill $\frac{\partial}{\partial s} f(\gamma(s)) = const.$ *geodesic* functions. They correspond to linear maps in Euclidean space and encode a constant variation with respect to the geodesic distance of the manifold. It is however possible that apart from the trivial case $f = const.$ no other geodesic functions exist on $M$. What is the implication of these results for regression? First, the use of Laplacian regularization leads always to a bias towards the constant function and does not extrapolate beyond data points. On the contrary, Hessian regularization is not biased towards constant functions if geodesic functions exist and extrapolates "linearly" (if possible) beyond data points. These crucial differences are illustrated in Figure 1 where we compare Laplacian regularization using the graph Laplacian as in [2] to Hessian regularization as introduced in the next section for a densely sampled spiral. Since the spiral is isometric to a subset of $\mathbb{R}$, it allows "geodesic" functions.

## 3 Semi-supervised regression using Hessian energy

As discussed in the last section unlabeled data provides us valuable information about the data manifold. We use this information to construct normal coordinates around each unlabeled point, which requires the estimation of the local structure of the manifold. Subsequently, we employ the normal coordinates to estimate the Hessian regularizer using the simple form of the second covariant derivative provided in Equation (1). It turns out that these two parts of our construction are similar to the one done in Hessian eigenmaps [3]. However, their estimate of the regularizer has stability problems when applied to semi-supervised regression as is discussed below. In contrast, the proposed method does not suffer from this short-coming and leads to significantly better performance. The solution of the semi-supervised regression problem is obtained by solving a sparse linear system. In the following, capital letters $X_i$ correspond to sample points and $x_r$ denote normal coordinates.

**Construction of local normal coordinates.** The estimation of local normal coordinates can be done using the set of $k$ nearest neighbors (NN) $N_k(X_i)$ of point $X_i$. The cardinality $k$ will be chosen later on by cross-validation. In order to estimate the local tangent space $T_{X_i}M$ (seen as an

$m$-dimensional affine subspace of $\mathbb{R}^d$), we perform PCA on the points in $N_k(X_i)$. The $m$ leading eigenvectors then correspond to an orthogonal basis of $T_{X_i}M$. In the ideal case, where one has a densely sampled manifold, the number of dominating eigenvalues should be equal to the dimension $m$. However, for real-world datasets like images the sampling is usually not dense enough so that the dimension of the manifold can not be detected automatically. Therefore the number of dimensions has to be provided by the user using prior knowledge about the problem or alternatively, and this is the way we choose in this paper, by cross-validation.

Having the exact tangent space $T_{X_i}M$ one can determine the normal coordinates $x_r$ of a point $X_j \in N_k(X_i)$ as follows. Let $\{u_r\}_{r=1}^m$ be the $m$ leading PCA eigenvectors, which have been normalized, then the normal coordinates $\{x_r\}_{r=1}^m$ of $X_j$ are given as,

$$x_r(X_j) = \langle u_r, X_j - X_i \rangle \frac{d_M(X_j, X_i)^2}{\sum_{r=1}^m \langle u_r, X_j - X_i \rangle^2},$$

where the first term is just the projection of the difference vector, $X_j - X_i$, on the basis vector $u_r \in T_{X_i}M$ and the second component is just a rescaling to fulfill the property of normal coordinates that the distance of a point $X_j \in M$ to the origin (corresponding to $X_i$) is equal to the geodesic distance $d_M(X_j, X_i)$ of $X_j$ to $X_i$ on $M$, $\|x(X_j)\|^2 = \sum_{r=1}^m x_r(X_i)^2 = d_M(X_j, X_i)^2$. The rescaling makes sense only if *local* geodesic distances can be accurately estimated. In our experiments, this was only the case for the 1D-toy dataset of Figure 1. For all other datasets we therefore use $x_r(X_j) = \langle u_r, X_j - X_i \rangle$ as normal coordinates. In [11] it is shown that this replacement yields an error of order $O(\|\nabla_a f\|^2 \kappa^2)$ in the estimation of $\|\nabla_a \nabla_b f\|^2$, where $\kappa$ is the maximal principal curvature (the curvature of $M$ with respect to the ambient space $\mathbb{R}^d$).

**Estimation of the Hessian energy.** The Hessian regularizer, the squared norm of the second covariant derivative, $\|\nabla_a \nabla_b f\|^2$, corresponds to the Frobenius norm of the Hessian of $f$ in normal coordinates, see Equation 1. Thus, given normal coordinates $x_r$ at $X_i$ we would like to have an operator $H$ which given the function values $f(X_j)$ on $N_k(X_i)$ estimates the Hessian of $f$ at $X_i$,

$$\frac{\partial^2 f}{\partial x_r \partial x_s}\Big|_{X_i} \approx \sum_{j=1}^k H_{rsj}^{(i)} f(X_j).$$

This can be done by fitting a second-order polynomial $p(x)$ in normal coordinates to $\{f(X_j)\}_{j=1}^k$,

$$p^{(i)}(x) = f(X_i) + \sum_{r=1}^n B_r x_r + \sum_r^n \sum_{s=r}^n A_{rs} x_r x_s, \tag{2}$$

where the zeroth-order term is fixed at $f(X_i)$. In the limit as the neighborhood size tends to zero, $p^{(i)}(x)$ becomes the second-order Taylor expansion of $f$ around $X_i$, that is,

$$B_r = \frac{\partial f}{\partial x_r}\Big|_{X_i}, \qquad A_{rs} = \frac{1}{2}\frac{\partial^2 f}{\partial x_r \partial x_s}\Big|_{X_i}, \tag{3}$$

with $A_{rs} = A_{sr}$. In order to fit the polynomial we use standard linear least squares,

$$\underset{w \in \mathbb{R}^P}{\arg\min} \sum_{j=1}^k \left( \left( f(X_j) - f(X_i) \right) - (\Phi w)_j \right)^2,$$

where $\Phi \in \mathbb{R}^{k \times P}$ is the design matrix with $P = m + \frac{m(m+1)}{2}$. The corresponding basis functions $\phi$, are the monomials, $\phi = [x_1, \ldots, x_m, x_1 x_1, x_1 x_2, \ldots, x_m x_m]$, of the normal coordinates (centered at $X_i$) of $X_j \in N_k(x_i)$ up to second order. The solution $w \in \mathbb{R}^P$ is $w = \Phi^+ f$, where $f \in \mathbb{R}^k$ and $f_j = f(X_j)$ with $X_j \in N_k(X_i)$ and $\Phi^+$ denotes the pseudo-inverse of $\Phi$.

Note, that the last $\frac{m(m+1)}{2}$ components of $w$ correspond to the coefficients $A_{rs}$ of the polynomial (up to rescaling for the diagonal components) and thus with Equation (3) we obtain the desired form $H_{rsj}^{(i)}$. An estimate of the Frobenius norm of the Hessian of $f$ at $X_i$ is thus given as,

$$\|\nabla_a \nabla_b f\|^2 \approx \sum_{r,s=1}^m \left( \sum_{\alpha=1}^k H_{rs\alpha}^{(i)} f_\alpha \right)^2 = \sum_{\alpha,\beta=1}^k f_\alpha f_\beta B_{\alpha\beta}^{(i)},$$

where $B_{\alpha\beta}^{(i)} = \sum_{r,s=1}^{m} H_{rs\alpha}^{(i)} H_{rs\beta}^{(i)}$ and finally the total estimated Hessian energy $\hat{S}_{\text{Hess}}(f)$ is the sum over all data points, where $n$ denotes the number of unlabeled and labeled points,

$$\hat{S}_{\text{Hess}}(f) = \sum_{i=1}^{n} \sum_{r,s=1}^{m} \left( \frac{\partial^2 f}{\partial x_r \partial x_s}\Big|_{X_i} \right)^2 = \sum_{i=1}^{n} \sum_{\alpha \in N_k(X_i)} \sum_{\beta \in N_k(X_i)} \mathbf{f}_\alpha \mathbf{f}_\beta B_{\alpha\beta}^{(i)} = \langle f, Bf \rangle \,,$$

where $B$ is the accumulated matrix summing up all the matrices $B^{(i)}$. Note, that $B$ is sparse since each point $X_i$ has only contributions from its neighbors.

Moreover, since we sum up the energy over all points, the squared norm of the Hessian is actually weighted with the local density of the points leading to a stronger penalization of the Hessian in densely sampled regions. The same holds for the estimate $\hat{S}_\Delta(f)$ of Laplacian regularization, $\hat{S}_\Delta(f) = \sum_{i,j=1}^{n} w_{ij}(f_i - f_j)^2$, where one also sums up the contributions of all data points (the rigorous connection between $\hat{S}_\Delta(f)$ and $S_\Delta(f)$ has been established in [2, 5]).

The effect of non-uniform sampling can be observed in Figure 1. There the samples of the spiral are generated by uniform sampling of the angle leading to a more densely sampled "interior" region, which leads to the non-linear behavior of the function for the Laplacian regularization. For the Hessian energy this phenomena cannot be seen in this example, since the Hessian of a "geodesic" function is zero everywhere and therefore it does not matter if it is weighted with the density. On the other hand for non-geodesic functions the weighting matters also for the Hessian energy. We did not try to enforce a weighting with respect to the uniform density. However, it would be no problem to compensate the effects of non-uniform sampling by using a weighted from of the Hessian energy.

**Final algorithm.** Using the ideas of the previous paragraphs the final algorithmic scheme for semi-supervised regression can now be immediately stated. We have to solve,

$$\underset{f \in \mathbb{R}^n}{\arg\min} \; \frac{1}{l} \sum_{i=1}^{l} (Y_i - f(X_i))^2 + \lambda \langle f, Bf \rangle \,, \tag{4}$$

where for notational simplicity we assume that the data is ordered such that the first $l$ points are labeled. The solution is obtained by solving the following sparse linear system,

$$(\mathbb{I}' + l\,\lambda B)f = Y,$$

where $\mathbb{I}'$ is the diagonal matrix with $\mathbb{I}'_{ii} = 1$ if $i$ is labeled and zero else and $Y_i = 0$ if $i$ is not labeled. The sparsity structure of $B$ is mainly influencing the complexity to solve this linear system. However, the number of non-zeros entries of $B$ is between $O(nk)$ and $O(nk^2)$ depending on how well behaved the neighborhoods are (the later case corresponds basically to random neighbors) and thus grows linearly with the number of data points.

**Stability of estimation procedure of Hessian energy.** Since we optimize the objective in Equation (4) for *any* possible assignment of function values $f$ on the data points, we have to ensure that the estimation of the Hessian is accurate for any possible function. However, the quality of the estimate of the Hessian energy depends on the quality of the local fit of $p^{(i)}$ for each data point $X_i$. Clearly, there are function assignments where the estimation goes wrong. If $k < P$ ($P$ is the number of parameters of the polynomial) $p$ can overfit the function and if $k > P$ then $p$ generally underfits. In both cases, the Hessian estimation is inaccurate. Most dangerous are the cases where the norm of the Hessian is underestimated in particular if the function is heavily oscillating. Note that during the estimation of local Hessian, we do not use the full second-order polynomial but fix its zeroth-order term at the value of $f$ (i.e. $p^{(i)}(X_i) = f(X_i)$; cf. Eq. (2)). The reason for this is that underfitting is much more likely if one fits a full second-order polynomial since the additional flexibility in fitting the constant term always reduces the Hessian estimate. In the worst case a function which is heavily oscillating can even have zero Hessian energy, if it allows a linear fit at each point, see Figure 3. If such a function fits the data well we get useless regression results[1] see Fig. 3. While fixing the constant term does not completely rule out such undesired behavior, we did not observe such irregular solutions in any experiment. In the appendix we discuss a modification of (Eq. (4)) which rules out

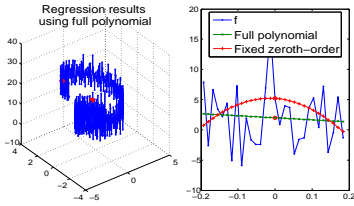
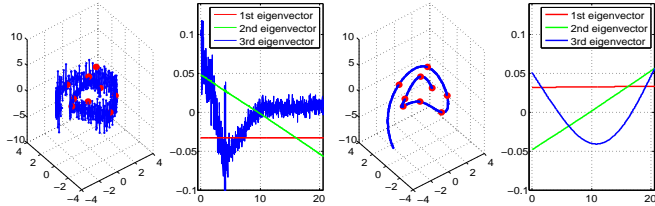

Figure 2: Fitting two points on the spiral revisited (see Fig. 1): Left image shows the regression result $f$ using the Hessian energy estimated by fitting a full polynomial in normal coordinates. The Hessian energy of this heavily oscillating function is 0, since every local fit is linear (an example shown in the right image; green curve). However, fixing the zeroth-order term yields a high Hessian energy as desired (local fit is shown as the red curve in the right image).

Figure 3: Sinusoid on the spiral: Left two images show the result of semi-supervised regression using the Hessian estimate of [3] and the corresponding smallest eigenvectors of the Hessian "matrix". One observes heavy oscillations, due to the bad estimation of the Hessian. The right two images show the result of our method. Note, that in particular the third eigenvector corresponding to a non-zero eigenvalue of $B$ is much better behaved.

for sure irregular solutions, but since it did not lead to significantly better experimental results and requires an additional parameter to tune we do not recommend to use it.

Our estimation procedure of the Hessian has similar motivation as the one done in Hessian eigenmaps [3]. However, in their approach they do not fix the zeroth-order term. This seems to be suitable for Hessian eigenmaps as they do not use the full Hessian, but only its $m+1$-dimensional null space (where $m$ is the intrinsic dimension of the manifold). Apparently, this resolves the issues discussed above so that the null space can still be well estimated also with their procedure. However, using their estimator for semi-supervised regression leads to useless results, see Fig. 3. Moreover, we would like to note that using our estimator not only the eigenvectors of the null space but also eigenvectors corresponding to higher eigenvalues can be well estimated, see Fig. 3.

## 4 Experiments

We test our semi-supervised regression method using Hessian regularization on one synthetic and two real-world data sets. We compare with the results obtained using Laplacian-based regularization and kernel ridge regression (KRR) trained only with the labeled examples. The free parameters for our method are the number of neighbors $k$ for k-NN, the dimensionality of the PCA subspace, and the regularization parameter $\lambda$ while the parameters for the Laplacian regularization-based regression are: $k$ for k-NN, the regularization parameter and the width of the Gaussian kernel. For KRR we used also the Gaussian kernel with the width as free parameter. These parameters were chosen for each method using 5-fold cross-validation on the labeled examples. For the digit and figure datasets, the experiments were repeated with 5 different assignments of labeled examples.

**Digit Dataset.** In the first set of experiments, we generated 10000 random samples of artificially generated images (size $28 \times 28$) of the digit 1. There are four variations in the data: translation (two variations), rotation and line thickness. For this dataset we are doing semi-supervised dimensionality reduction since the task is to estimate the natural parameters which were used to generate the digits. This is done based on 50 and 100 labeled images. Each of the variation corresponds then to a separate regression problem which we finally stick together to get an embedding into four dimensions. Note, that this dataset is quite challenging since translation of the digit leads to huge Euclidean distances between digits although they look visually very similar. Fig. 2 and Table 1 summarize the results. As observed in the first row of Fig. 2, KRR ($K$) and Hessian ($H$) regularization recover well the two parameters of line width and rotation (all other embeddings can be found in the supplementary material). As discussed previously, the Laplacian ($L$) regularization tends to shrink the estimated parameters towards a constant as it penalizes the "geodesic" functions. This results in the poor estimation of parameters, especially the line-thickness parameter.[2] Although KRR estimates well the thickness parameter, it fails for the rotation parameter (cf. the second row of Fig. 2 where we

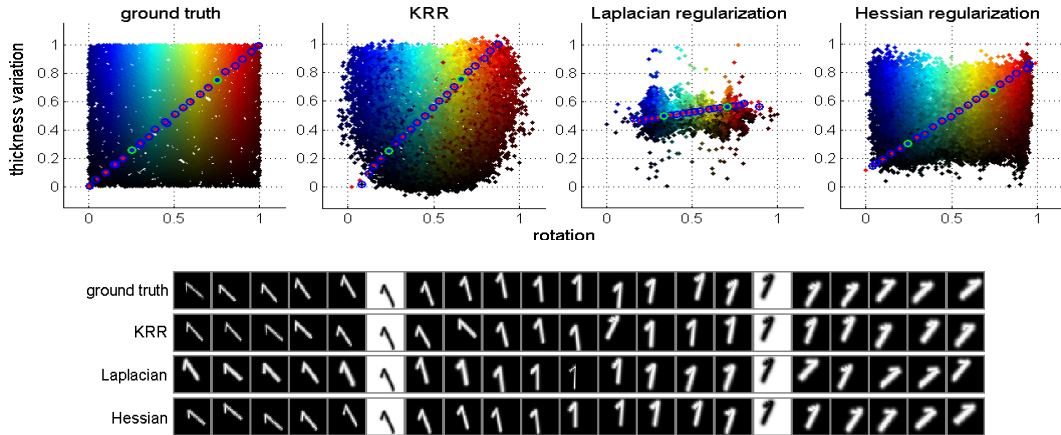

Figure 2: Results on the digit 1 dataset. First row: the 2D-embedding of the digits obtained by regression for the rotation and thickness parameter with 100 labels. Second row: 21 digit images sampled at regular intervals in the estimated parameter spaces: two reference points (inverted images) are sampled in the ground truth parameter space and then in the corresponding estimated embedding. Then, 19 points are sampled in the estimated parameter spaces based on linear inter/extrapolation of the parameters. The shown image samples are the ones which have parameters closest to the interpolated ones. In each parameter space the interpolated points, the corresponding closest data points and the reference points are marked with red dots, blue and cyan circles.

Table 1: Results on digits: mean squared error (standard deviation) (both in units $10^{-3}$).

|  | 50 labeled points | | | | 100 labeled points | | | |
| --- | --- | --- | --- | --- | --- | --- | --- | --- |
|  | h-trans. | v-trans. | rotation | thickness | h-trans. | v-trans. | rotation | thickness |
| $K$ | 0.78(0.13) | 0.85(0.14) | 45.49(7.20) | 0.02(0.01) | 0.39(0.10) | 0.48(0.08) | 26.02(2.98) | 0.01(0.00) |
| $L$ | 2.41(0.26) | 3.91(0.59) | 64.56(3.90) | 0.39(0.02) | 1.17(0.13) | 2.20(0.22) | 30.73(6.05) | 0.34(0.01) |
| $H$ | 0.34(0.03) | 0.88(0.07) | 4.03(1.15) | 0.15(0.02) | 0.16(0.03) | 0.39(0.07) | 1.48(0.26) | 0.06(0.01) |

show the images corresponding to equidistant inter/extrapolation in the estimated parameter space between two fixed digits (inverted image)). The Hessian regularization provided a moderate level of accuracy in recovering the thickness parameter and performed best on the remaining ones.

**Figure Dataset.** The second dataset consists of 2500 views of a toy figure (see Fig. 3) sampled based on regular intervals in zenith and azimuth angles on the upper hemisphere around the centered object [10]. Fig. 3 shows the results of regression for three parameters - the zenith angle, and the azimuth angle is transformed into Euclidean x,y coordinates.[3] Both Laplacian and Hessian regularizers provided significantly better estimation of the parameters in comparison to KRR, which demonstrates the effectiveness of semi-supervised regression. However, the Laplacian shows again contracting behavior which is observed in the top view of hemisphere. Note that for our method this does not occur and the spacing of the points in the parameter space is much more regular, which again stresses the effectiveness of our proposed regularizer.

**Image Colorization.** Image colorization refers to the task of estimating the color components of a given gray level image. Often, this problem is approached based on the color information of a subset of pixels in the image, which is specified by a user (cf. [8] for more details). This is essentially a semi-supervised regression problem where the user-specified color components correspond to the labels. To facilitate quantitative evaluation, we adopted 20 color images, sampled a subset of pixels in each image as labels, and used the corresponding gray levels as inputs. The number of labeled points were 30 and 100 for each images, which we regard as a moderate level of user intervention. As error measure, we use the mean square distance between the original image and the corresponding

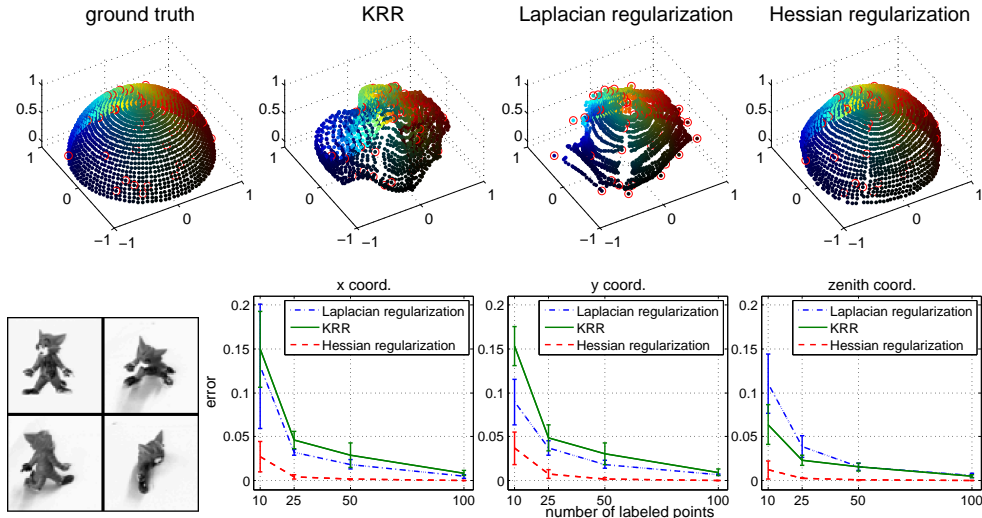

Figure 3: Results of regression on the figure dataset. First row: embedding in the three dimensional spaces with 50 labels. Second row: Left: some example images of the dataset, Right: error plots for each regression variable for different number of labeled points.

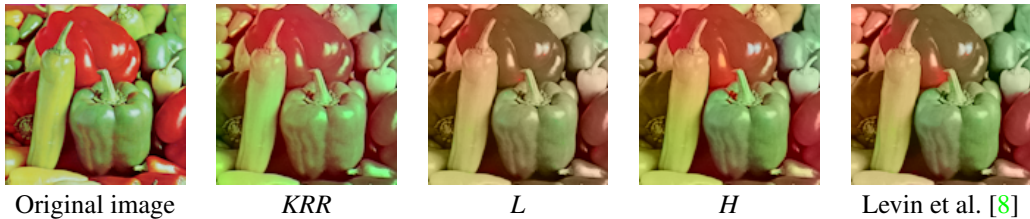

| Original image | *KRR* | *L* | *H* | Levin et al. [8] |

Figure 4: Example of image colorization with 30 labels. KRR failed in reconstructing (the color of) the red pepper at the lower-right corner, while the Laplacian regularizer produced overall, a greenish image. Levin et al's method well-recovered the lower central part however failed in reconstructing the upper central pepper. Despite the slight diffusion of red color at the upper-left corner, overall, the result of Hessian regularization looks best which is also confirmed by the reconstruction error.

reconstruction in the RGB space. During the colorization, we go over to the YUV color model such that the Y components, containing the gray level values, are used as the input, based on which the U and V components are estimated. The estimated U-V components are then combined with the Y component and converted into RGB format. For the regression, for each pixel, we use as features the $3 \times 3$-size image patch centered at the pixel of interest plus the 2-dimensional coordinate value of that pixel. The coordinate values are weighted by 10 such that the contribution of coordinate values and gray levels is balanced. For comparison, we performed experiments with the method of Levin et al. [8] as one of the state-of-the-art methods.[4] Figure 4 shows an example and Table 2 summarizes the results. The Hessian regularizer clearly outperformed the KRR and the Laplacian-based regression and produced slightly better results than those of Levin et al. [8]. We expect that the performance can be further improved by exploiting a priori knowledge on structure of natural images (e.g., by exploiting the segmentation information (cf. [9, 6]) in the NN structure).

Table 2: Results on colorization: mean squared error (standard deviation) (both in units $10^{-3}$).

| # labels | *K* | *L* | *H* | Levin et al. [8] |
|---|---|---|---|---|
| 30 | 1.18(1.10) | 0.83(0.64) | 0.64(0.50) | 0.74(0.61) |
| 100 | 0.66(0.65) | 0.50(0.33) | 0.32(0.25) | 0.37(0.26) |

## Footnotes

[1] For the full second-order polynomial even cross-validation does not rule out these irregular solutions.

[2]In this figure, each parameter is normalized to lie in the unit interval while the regression was performed in the original scale. The point $(0.5, 0.5)$ corresponds roughly to the origin in the original parameters.

[3]Although the underlying manifold is two dimensional, the parametrization cannot be directly found based on regression as the azimuth angle is periodic. This results in contradicting assignments of ground truth labels.

[4]Code is available at: http://www.cs.huji.ac.il/~yweiss/Colorization/.

# References

[1] M. Belkin and P. Niyogi. Laplacian eigenmaps for dimensionality reduction and data representation. *Neural Computation*, 15(6):1373–1396, 2003. 2

[2] M. Belkin and P. Niyogi. Semi-supervised learning on manifolds. *Machine Learning*, 56:209–239, 2004. 1, 3, 5

[3] D. Donoho and C. Grimes. Hessian eigenmaps: Locally linear embedding techniques for high-dimensional data. *Proc. of the National Academy of Sciences*, 100(10):5591–5596, 2003. 2, 3, 6

[4] J. Eells and L. Lemaire. *Selected topics in harmonic maps*. AMS, Providence, RI, 1983. 3

[5] M. Hein. Uniform convergence of adaptive graph-based regularization. In G. Lugosi and H. Simon, editors, *Proc. of the 19th Conf. on Learning Theory (COLT)*, pages 50–64, Berlin, 2006. Springer. 5

[6] R. Irony, D. Cohen-Or, and D. Lischinski. Colorization by example. In *Proc. Eurographics Symposium on Rendering*, pages 201–210, 2005. 8

[7] J. M. Lee. *Riemannian Manifolds - An introduction to curvature*. Springer, New York, 1997. 2

[8] A. Levin, D. Lischinski, and Y. Weiss. Colorization using optimization. In *Proc. SIGGRAPH*, pages 689–694, 2004. 7, 8

[9] Q. Luan, F. Wen, D. Cohen-Or, L. Liang, Y.-Q. Xu, and H.-Y. Shum. Natural image colorization. In *Proc. Eurographics Symposium on Rendering*, pages 309–320, 2007. 8

[10] G. Peters. Efficient pose estimation using view-based object representations. *Machine Vision and Applications*, 16(1):59–63, 2004. 7

[11] F. Steinke and M. Hein. Non-parametric regression between Riemannian manifolds. In *Advances in Neural Information Processing Systems*, pages 1561–1568, 2009. 2, 4

[12] J. J. Verbeek and N. Vlassis. Gaussian fields for semi-supervised regression and correspondence learning. *Pattern Recognition*, 39:1864–1875, 2006. 1, 3

[13] X. Yang, H. Fu, H. Zha, and J. Barlow. Semi-supervised nonlinear dimensionality reduction. In *Proc. of the 23rd international conference on Machine learning*, pages 1065–1072, New York, NY, USA, 2006. ACM. 1

